# DTs: Dynamic Trees

**Christopher K. I. Williams**     **Nicholas J. Adams**
Institute for Adaptive and Neural Computation
Division of Informatics, 5 Forrest Hill
Edinburgh, EH1 2QL, UK.     http://www.anc.ed.ac.uk/
ckiw@dai.ed.ac.uk            nicka@dai.ed.ac.uk

## Abstract

In this paper we introduce a new class of image models, which we call dynamic trees or DTs. A dynamic tree model specifies a prior over a large number of trees, each one of which is a tree-structured belief net (TSBN). Experiments show that DTs are capable of generating images that are less blocky, and the models have better translation invariance properties than a fixed, "balanced" TSBN. We also show that Simulated Annealing is effective at finding trees which have high posterior probability.

## 1 Introduction

In this paper we introduce a new class of image models, which we call dynamic trees or DTs. A dynamic tree model specifies a prior over a large number of trees, each one of which is a tree-structured belief net (TSBN). Our aim is to retain the advantages of tree-structured belief networks, namely the hierarchical structure of the model and (in part) the efficient inference algorithms, while avoiding the "blocky" artifacts that derive from a single, fixed TSBN structure. One use for DTs is as prior models over labellings for image segmentation problems.

Section 2 of the paper gives the theory of DTs, and experiments are described in section 3.

## 2 Theory

There are two essential components that make up a dynamic tree network (i) the tree architecture and (ii) the nodes and conditional probability tables (CPTs) in the given tree. We consider the architecture question first.

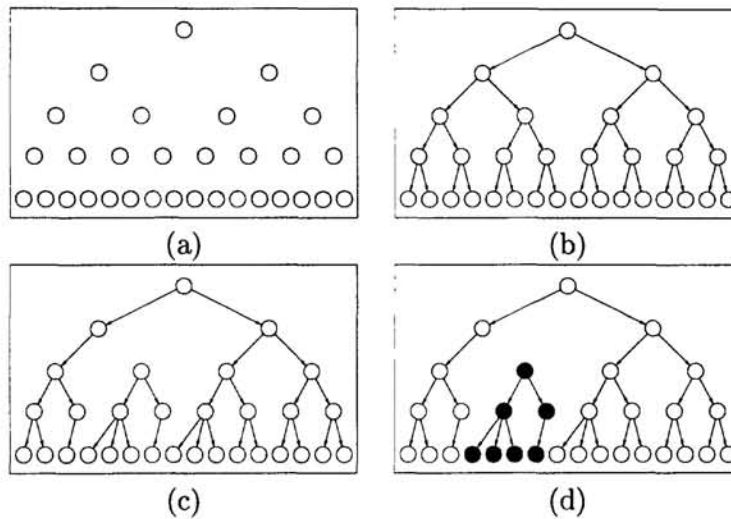

Figure 1: (a) "Naked" nodes, (b) the "balanced" tree architecture, (c) a sample from the prior over $Z$, (d) data generated from the tree in (c).

Consider a number of nodes arranged into layers, as in Figure 1(a). We wish to construct a tree structure so that any child node in a particular layer will be connected to a parent in the layer above. We also allow there to be a null parent for each layer, so that any child connected to it will become a new root. (Technically we are constructing a forest rather than a tree.) An example of a structure generated using this method is shown in Figure 1(c).

There are a number of ways of specifying a prior over trees. If we denote by $z_i$ the indicator vector which shows to which parent node $i$ belongs, then the tree structure is specified by a matrix $Z$ whose columns are the individual $z_i$ vectors (one for each node). The scheme that we have investigated so far is to set $P(Z) = \prod_i P(z_i)$.

In our work we have specified $P(z_i)$ as follows. Each child node is considered to have a "natural" parent—its parent in the balanced structure shown in Figure 1(b). Each node in the parent layer is assigned an "affinity" for each child node, and the "natural" parent has the highest affinity. Denote the affinity of node $k$ in the parent layer by $a_k$. Then we choose $P(z_i = e_k) = e^{\beta a_k} / \sum_{j \in Pa_i} e^{\beta a_j}$, where $\beta$ is some positive constant and $e_k$ is the unit vector with a 1 in position $k$. Note that the "null" parent is included in the sum, and has affinity $a_{null}$ associated with it, which affects the relative probability of "orphans". We have named this prior the "full-time-node-employment" prior as all the nodes participate in the creation of the tree structure to some degree.

Having specified the prior over architectures, we now need to translate this into a TSBN. The units in the tree are taken to be $C$-class multinomial random variables. Each *layer* of the structure has associated with it a prior probability vector $\pi_l$ and CPT $M_l$. Given a particular $Z$ matrix which specifies a forest structure, the probability of a particular instantiation of all of the random variables is simply the product of the probabilities of all of the trees, where the appropriate root probabilities and CPTs are picked up from the $\pi_l$s and $M_l$s. A sample generated from the tree structure in Figure 1(c) is shown in Figure 1(d).

Our intuition as to why DTs may be useful image models is based on the idea that most pixels in an image are derived from a single object. We think of an object as being described by a root of a tree, with the scale of the object being determined by the level in the tree at which the root occurs. In this interpretation the CPTs will have most of their probability mass on the diagonal.

Given some data at the bottom layer of units, we can form a posterior over the tree structures and node instantiations of the layers above. This is rather like obtaining a set of parses for a number of sentences using a context-free grammar[1].

In the DT model as described above different examples are explained by different trees. This is an important difference with the usual priors over belief networks as used, e.g. in Bayesian averaging over model structures. Also, in the usual case of model averaging, there is normally no restriction to TSBN structures, or to tying the parameters ($\pi_l$s and $M_l$s) between different structures.

## 2.1  Inference in DTs

We now consider the problem of inference in DTs, i.e. obtaining the posterior $P(Z, X_h|X_v)$ where $Z$ denotes the tree-structure, $X_v$ the visible units (the image clamped on the lowest level) and $X_h$ the hidden units. In fact, we shall concentrate on obtaining the posterior marginal $P(Z|X_v)$, as we can obtain samples from $P(X_h|X_v, Z)$ using standard techniques for TSBNs.

There are a very large number of possible structures; in fact for a set of nodes created from a balanced tree with branching factor $b$ and depth $D$ (with the top level indexed by 1) there are $\prod_{d=2}^{D}(b^{(d-2)} + 1)^{b^{(d-1)}}$ possible forest structures. Our objective will be to obtain the maximum a posteriori (MAP) state from the posterior $P(Z|X_v) \propto P(Z)P(X_v|Z)$ using Simulated Annealing.[2] This is possible because two components $P(Z)$ and $P(X_v|Z)$ are readily evaluated. $P(X_v|Z)$ can be computed from $\prod_r(\sum_{x_r} \lambda(x_r)\pi(x_r))$, where $\lambda(x_r)$ and $\pi(x_r)$ are the Pearl-style vectors of each root $r$ of the forest.

An alternative to sampling from the posterior $P(Z, X_h|X_v)$ is to use approximate inference. One possibility is to use a mean-field-type approximation to the posterior of the form $Q_Z(Z)Q_h(X_h)$ (Zoubin Ghahramani, personal communication, 1998).

## 2.2  Comparing DTs to other image models

Fixed-structure TSBNs have been used by a number of authors as models of images (Bouman and Shapiro, 1994), (Luettgen and Willsky, 1995). They have an attractive multi-scale structure, but suffer from problems due to the fixed tree structure, which can lead to very "blocky" segmentations. Markov Random Field (MRF) models are also popular image models; however, one of their main limitations is that inference in a MRF is NP-hard. Also, they lack an hierarchical structure. On the other hand, stationarity of the process they define can be easily ensured, which

is not the case for fixed-structure TSBNs. One strategy to overcome the fixed structure of TSBNs is to break away from the tree structure, and use belief networks with cross connections e.g. (Dayan *et al.*, 1995). However, this means losing the linear-time belief-propagation algorithms that can be used in trees (Pearl, 1988) and using approximate algorithms. While it is true that inference over DTs is also NP-hard, we do retain a"clean" semantics based on the fact that we expect that each pixel should belong to one object, which may lead to useful approximation schemes.

## 3  Experiments

In this section we describe two experiments conducted on the DT models. The first has been designed to compare the translation performance of DTs with that of the balanced TSBN structure and is described in section 3.1. In section 3.2 we generate 2-d images from the DT model, find the MAP Dynamic Tree for these images, and contrast their performance in relative to the balanced TSBN.

### 3.1  Comparing DTs with the balanced TSBN

We consider a 5-layer binary tree with 16 leaf nodes, as shown in Figure 1. Each node in the tree is a binary variable, taking on values of white/black. The $\pi_l$'s, $M_l$'s and affinities were set to be equal in each layer. The values used were $\pi = (0.75, 0.25)$ with 0.75 referring to white, and $M$ had values 0.99 on the diagonal and 0.01 off-diagonal. The affinities[3] were set as 1 for the natural parent, 0 for the nearest neighbour(s) of the natural parent, $-\infty$ for non-nearest neighbours and $a_{null} = 0$, with $\beta = 1.25$.

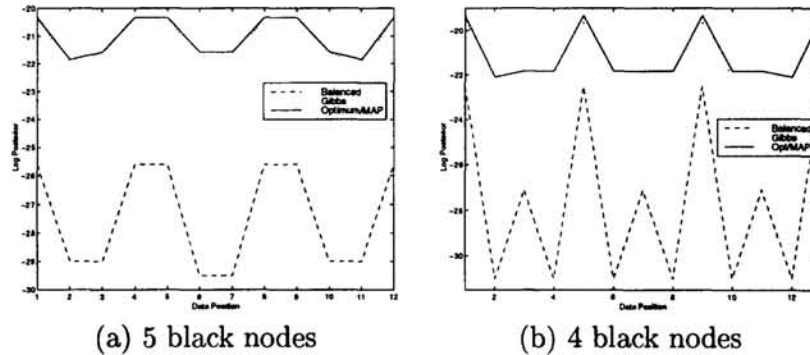

(a) 5 black nodes          (b) 4 black nodes

Figure 2: Plots of the unnormalised log posterior vs position of the input pattern for (a) the 5-black-nodes pattern and (b) 4-black-nodes pattern.

To illustrate the effects of translation, we have taken a stimulus made up of a bar of five black pixels, and moved it across the image. The unnormalised log posterior for a particular $Z$ configuration is $\log P(Z) + \log P(X_v|Z)$. This is computed for the balanced TSBN architecture, and compared to the highest value that can be found by conducting a search over $Z$. These results are plotted in Figure 2(a). The $x$-axis denotes the position of the left hand end of the bar (running from 1 to

12), and the $y$-axis shows the posterior probability. Note that due to symmetries there are in reality fewer than 12 distinct configurations. Figure 2(a) shows clearly that the balanced TSBN is a poor model for this stimulus, and that much better interpretations can be found using DTs, even though the "natural parent" idea ensures that the $\log P(Z)$ is always larger for the balanced tree.

Notice also how the balanced TSBN displays greater sensitivity of the log posterior with respect to position than the DT model. Figure 2 shows both the "optimal" log posterior (found "by hand", using intuitions as to the best trees), and the those of the MAP models discovered by Simulated Annealing. Annealing was conducted from a starting temperature of 1.0 and exponentially decreased by a factor of 0.9. At each temperature up to 2000 proposals could be made, although transition to the next temperature would occur after 200 accepted steps. The run was deemed to have converged after five successive temperature steps were made without accepting a single step. We also show the log posterior of trees found by Gibbs sampling from which we report the best configuration found from four separate runs (with different random starting positions), each of which was run for 25,000 sweeps through all of the nodes.

In Figure 2(b) we have shown the log posterior for a stimulus made up of *four* black nodes[4]. In this case the balanced TSBN is even more sensitive to the stimulus location, as the four black nodes fit exactly under one sub-tree when they are in positions 1, 5, 9 or 13. By contrast, the dynamic tree is less sensitive to the alignment, although it does retain a preference for the configuration most favoured by the balanced TSBN. This is due to the concept of a "natural" parent built into the (current) architecture (but see Section 4 for further discussion).

Clearly these results are somewhat sensitive to settings of the parameters. One of the most important parameters is the diagonal entry in the CPT. This controls the relative desirability of having a disconnection against a transition in the tree that involves a colour change. For example, if the diagonal entry in the CPT is reduced to 0.95, the gap between the optimal and balanced trees in Figure 2(b) is decreased. We have experimented with CPT entries of 0.90, 0.95 and 0.99, but otherwise have not needed to explore the parameter space to obtain the results shown.

### 3.2 Generating from the prior and finding the MAP Tree in 2-d

We now turn our attention to 2-d images. Considering a 5 layer quad-tree node arrangement gives a total of 256 leaf nodes or a 16x16 pixel image. A structural plot of such a tree generated from the prior is shown in figure 3.

Each sub-plot is a slice through the tree showing the nodes on successive levels. The boxes represent a single node on the current level and their shading indicates the tree to which they belong. Nodes in the parent layer above are superimposed as circles and the lines emanating from them shows their connectivity. Black circles with a smaller white circle inside are used to indicate root nodes. Thus in the example above we see that the forest consists of five trees, four of whose roots lie at level 3 (which between them account for most of the black in the image, Figure 3(f)), while the root node at level 1 is responsible for the background.

[4]The parameters are the same as above, except that $a_{null}$ in level 3 was set to 10.0 to encourage disconnections at this level.

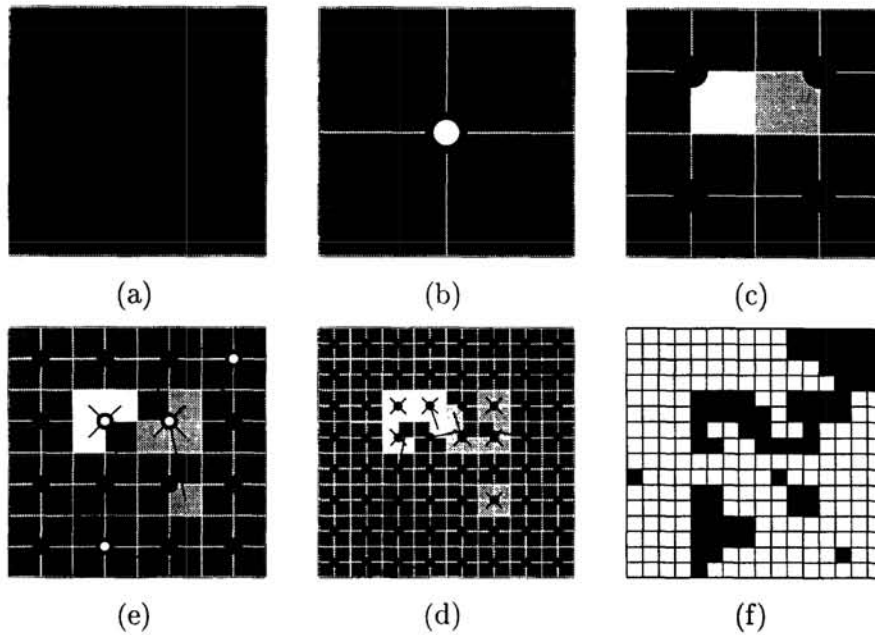

(a)   (b)   (c)

(e)   (d)   (f)

Figure 3: Plot of the MAP Dynamic Tree of the accompanying image (f).

Broadly speaking the parameters for the 2-d DTs were set to be similar to the 1-d trees of the previous section, except that the disconnection affinities were set to favour disconnections higher up the tree, and to values for the leaf level such that leaf disconnection probabilities tend to zero. In practice this resulted in all leaves being connected to parent nodes (which is desirable as we believe that single-pixel objects are unlikely). The $\beta$ values increase with tree depth so that lower levels nodes choose parents from a tighter neighbourhood. The $\pi_l$ and $M_l$ values were unchanged, and again we consider binary valued nodes.

A suite of 600 images were created by sampling DTs from the above prior and then generating 5 images from each. Figure 3(f) shows an example of an image generated by the DT and it can be seen that the "blockiness" exhibited by balanced TSBNs is not present.

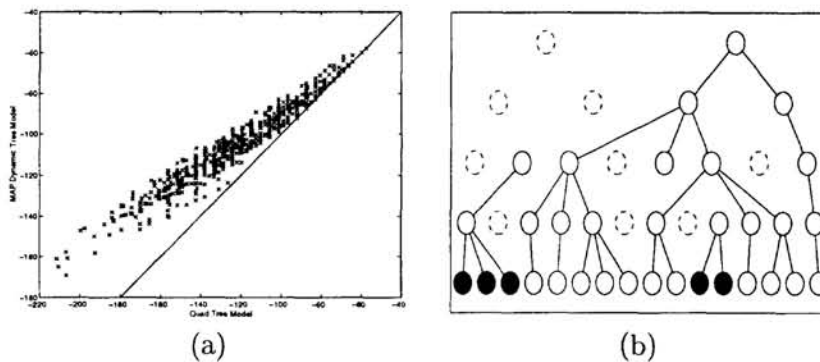

(a)   (b)

Figure 4: (a) Comparison of the MAP DT log posterior against that of the quad-tree for 600 images, (b) tree generated from the "part-time-node-employment" prior.

The MAP Dynamic Tree for each of these images was found by Simulated Annealing using the same exponential strategy described earlier, and their log posteriors are compared with those of the balanced TSBN in the plot 4(a). The line denotes the boundary of equal log posterior and the location of all the points above this clearly shows that in every case the MAP tree found has a higher posterior.

## 4    Discussion

Above we have demonstrated that DT models have greater translation invariance and do not exhibit the blockiness of the balanced TSBN model. We also see that Simulated Annealing methods are successful at finding trees that have high posterior probability.

We now discuss some extensions to the model. In the work above we have kept the balanced tree arrangement of nodes. However, this could be relaxed, giving rise to roughly equal numbers of nodes at the various levels (*cf stationary* wavelets). This would be useful (a) for providing better translation invariance and (b) to avoid slight shortages of hidden units that can occur when patterns that are "misaligned" wrt the balanced tree are presented. In this case the prior over $Z$ would need to be adjusted to ensure a high proportion of tree-like structures, by generating the $z$'s and $x$'s in layers, so that the $z$'s can be contingent on the states of the units in the layer above. We have devised a prior of this nature and called it the "part-time-employment" prior as the nodes can decide whether or not they wish to be employed in the tree structure or remain redundant and inactive. An example tree generated from this prior is shown in figure 4(b); we plan to explore this direction further in on-going research. Other research directions include the learning of parameters in the networks (e.g. using EM), and the introduction of additional information at the nodes; for example one might use real-valued variables in addition to the multinomial variables considered above. These additional variables might be used to encode information such as that concerning the instantiation parameters of objects.

### Acknowledgements

This work stems from a conversation between CW and Zoubin Gharahmani at the Isaac Newton Institute in October 1997. We thank Zoubin Ghahramani, Geoff Hinton and Peter Dayan for helpful conversations, and the Isaac Newton Institute for Mathematical Sciences (Cambridge, UK) for hospitality during the "Neural Networks and Machine Learning" programme. NJA is supported by an EPSRC research studentship, and the work of CW is partially supported by EPSRC grant GR/L03088, *Combining Spatially Distributed Predictions From Neural Networks*.

## Footnotes

[1]CFGs have a $O(n^3)$ algorithm to infer the MAP parse; however, this algorithm depends crucially on the one-dimensional ordering of the inputs. We believe that the possibility of crossed links in the DT architecture means that this kind of algorithm is not applicable to the DT case. Also, the DT model can be applied to 2-d images, where the $O(n^3)$ algorithm is not applicable.

[2]It is also possible to sample from the posterior using, e.g. Gibbs Sampling.

[3]The affinities are defined up to the addition of an arbitrary constant.

## References

Bouman, C. A. and M. Shapiro (1994). A Multiscale Random Field Model for Bayesian Image Segmentation. *IEEE Transactions on Image Processing* **3(2)**, 162–177.

Dayan, P., G. E. Hinton, R. M. Neal, and R. S. Zemel (1995). The Helmholtz Machine. *Neural Computation* **7(5)**, 889–904.

Luettgen, M. R. and A. S. Willsky (1995). Likelihood Calculation for a Class of Multiscale Stocahstic Models, with Application to Texture Discrimination. *IEEE Trans. Image Processing* **4(2))**, 194–207.

Pearl, J. (1988). *Probabilistic Reasoning in Intelligent Systems: Networks of Plausible Inference*. San Mateo, CA: Morgan Kaufmann.